# A Statistical Mechanics Approach to Approximate Analytical Bootstrap Averages

**Dörthe Malzahn**[(1)]          **Manfred Opper**[(2)]

[(1)] Informatics and Mathematical Modelling, Technical University of Denmark,
R.-Petersens-Plads Building 321, DK-2800 Lyngby, Denmark

[(2)] Neural Computing Research Group, School of Engineering and Applied Science,
Aston University, Birmingham B4 7ET, United Kingdom

dm@imm.dtu.dk          opperm@aston.ac.uk

## Abstract

We apply the replica method of Statistical Physics combined with a variational method to the approximate analytical computation of bootstrap averages for estimating the generalization error. We demonstrate our approach on regression with Gaussian processes and compare our results with averages obtained by Monte-Carlo sampling.

## 1  Introduction

The application of tools from Statistical Mechanics to analyzing the average case performance of learning algorithms has a long tradition in the Neural Computing and Machine Learning community [1, 2]. When data are generated from a highly symmetric distribution and the dimension of the data space is large, methods of statistical mechanics of disordered systems allow for the computation of learning curves for a variety of interesting and nontrivial models ranging from simple perceptrons to Support-vector Machines. Unfortunately, the specific power of this approach, which is able to give explicit *distribution dependent* results represents also a major drawback for *practical* applications. In general, data distributions are unknown and their replacement by simple model distributions might only reveal some qualitative behavior of the true learning performance.

In this paper we suggest a *novel application* of the Statistical Mechanics techniques to a topic within Machine Learning for which the distribution over data is well known and controlled by the experimenter. It is given by the resampling of an *existing* dataset in the so called *bootstrap* approach [3]. Creating *bootstrap* samples of the original dataset by random *resampling* with replacement and retraining the statistical model on the bootstrap sample is a widely applicable statistical technique. By replacing averages over the true *unknown* distribution of data with suitable averages over the bootstrap samples one can estimate various properties such as the bias, the variance and the generalization error of a statistical model.

While in general bootstrap averages can be approximated by Monte-Carlo sampling, it is useful to have also *analytical* approximations which avoid the time consuming retraining of the model for each sample. Existing analytical approximations (based on asymptotic techniques) such as the *delta* method and the *saddle point* method (see e.g.[5]) require

usually explicit analytical formulas for the estimators of the parameters for a trained model. These may not be easily obtained for more complex models in Machine Learning. In this paper, we discuss an application of the *replica method* of Statistical Physics [4] which combined with a variational method [6] can produce approximate averages over the random drawings of bootstrap samples. Explicit formulas for parameter estimates are avoided and replaced by the implicit condition that such estimates are expectations with respect to a certain Gibbs distribution to which the methods of Statistical Physics can be well applied. We demonstrate the method for the case of *regression with Gaussian processes (GP)* (which is a kernel method that has gained high popularity in the Machine Learning community in recent years [7]) and compare our analytical results with results obtained by Monte-Carlo sampling.

## 2 Basic setup and Gibbs distribution

We will keep the notation in this section fairly general, indicating that most of the theory can be developed for a broader class of models. We assume that a fixed set of data $D_0 = (z_1, z_2, \ldots, z_N)$ is modeled by a likelihood of the type

$$P(D|f) \propto \exp\left(-\sum_{j=1}^{N} h(f, z_j)\right) \tag{1}$$

where the "training error" $h$ is parametrized by a parameter $f$ (which can be a finite or even infinite dimensional object) which must be estimated from the data. We will later specialize to supervised learning problems where each data point $z = (x, y)$ consists of an input $x$ (usually a finite dimensional vector) and a real label $y$. In this case, $f$ stands for a function $f(x)$ which models the outputs, or for the parameters (like the weights of a neural network) which parameterize such functions. We will later apply our approach to the mean square error given by

$$h(f, z_j) = \frac{1}{2\sigma^2}(f(x_j) - y_j)^2 . \tag{2}$$

The first basic ingredient of our approach is the assumption that the estimator for the unknown "true" function $f$ can be represented as the mean with respect to a posterior distribution over all possible $f$'s. This avoids the problem of writing down explicit, complicated formulas for estimators. To be precise, we assume that the statistical estimator $\hat{f}_{D_0}$ (which is based on the training set $D_0$) can be represented as the expectation of $f$ with respect to the measure

$$p[f|D_0] = \frac{1}{Z}\mu[f] \exp\left(-\sum_{j=1}^{N} h(f, z_j)\right) \tag{3}$$

which is constructed from a suitable prior distribution $\mu[f]$ and the likelihood (1).

$$Z = \int d\mu[f] \exp\left(-\sum_{j=1}^{N} h(f, z_j)\right) \tag{4}$$

denotes a normalizing partition function. Our choice of (3) does not mean that we restrict ourselves to Bayesian estimators. By introducing specific ("temperature" like) parameters in the prior and the likelihood, the measure (3) can be strongly concentrated at its mean such that maximum likelihood/MAP estimators can be included in our framework.

## 3 Bootstrap averages

We will explain our analytical approximation to resampling averages for the case of supervised learning problems. If we are interested in, say, estimating the expected error on test

points [1] which are not contained in the training set $D_0$ of size $N$ and if we have no hold out data, we can create artificial data sets $D$ by resampling (with replacement) $m$ data from the original set $D_0$, where each data point $z_i \in D_0$ is taken with equal probability $1/N$. Hence, some of the $z_i$'s will appear several times in the *bootstrap* sample and others not at all. A proxy for the true average test error can be obtained by retraining the model on each bootstrap training set $D$, calculating the test error only on those points which are *not* contained in $D$ and finally averaging over many sets $D$. In practice, the case $m = N$ maybe of main importance, but we will also allow for estimating a lager part of the "learning curve" by allowing for $m < N$ and $m > N$. We will not discuss the statistical properties of such bootstrap estimates and their refinements (such as Efron's .632 estimate) in this paper, but refer the reader to the standard literature [3, 5].

For any given set $D_0$, we represent a bootstrap sample $D$ by the vector of "occupation" numbers $\mathbf{m} = (m_1, \ldots, m_N)$ with $\sum_{i=1}^{N} m_i = m$. $m_i$ is the number of times example $z_i$ appears in the set $D$. Denoting the expectation over random bootstrap samples by $E_D$, Efron's estimator for the bootstrap generalization error is

$$\varepsilon(m) \doteq \frac{1}{N} \sum_{i=1}^{N} \frac{E_D \left[ \delta_{m_i,0} \left( \hat{f}_D(x_i) - y_i \right)^2 \right]}{E_D \left[ \delta_{m_i,0} \right]} \tag{5}$$

where we specialized to the square error for testing. Eq.(5) computes the average bootstrap test error at each data point $i$ from $D_0$. The Kronecker symbol, defined by $\delta_{i,j} = 1$ for $i = j$ and 0 else, guarantees that only realizations of bootstrap training sets $D$ contribute which do not contain the test point. Introducing the abbreviation

$$\mathcal{L}(f, z_i) = f(x_i) - y_i \tag{6}$$

(which is a *linear* function of $f$), and using the definition of the estimator $\hat{f}_D$ as an average of $f$'s over the Gibbs distribution (3), the bootstrap estimate (5) can be rewritten as

$$\varepsilon(m) = \frac{1}{N} \sum_{i=1}^{N} \frac{1}{E_D \left[ \delta_{m_i,0} \right]} E_D \left[ \delta_{m_i,0} \frac{1}{Z^2} \int d\mu[f_1] d\mu[f_2] \, \mathcal{L}(f_1, z_i) \, \mathcal{L}(f_2, z_i) \times \right.$$

$$\left. \times \exp \left( - \sum_{j=1}^{N} m_j (h(f_1, z_j) + h(f_2, z_j)) \right) \right] . \tag{7}$$

which involves 2 copies (or *replicas*) $f_1$ and $f_2$ of the variable $f$. More complicated types of test errors which are polynomials or can be approximated by polynomials in $\hat{f}_D$ can be rewritten in a similar way, involving more replicas of the variable $f$.

## 4  Analytical averages using the "replica trick"

For fixed $m$, the distribution of $m_i$'s is multinomial. It is simpler (and does not make a big difference when $m$ is sufficiently large) when we work with a Poisson distribution for the size of the set $D$ with $m$ as the mean number of data points in the sample. In this case we get the simpler, factorizing joint distribution

$$P(\mathbf{m}) = \prod_{i=1}^{N} \frac{\mu^{m_i} e^{-\mu}}{m_i!} \tag{8}$$

for the occupation numbers $m_i$ where $\mu = m/N$. With Eq. (8) follows $E_D[\delta_{m_i,0}] = e^{-\mu}$.

To enable the analytical average over the vector $\mathbf{m}$ (which is the "quenched disorder" in the language of Statistical Physics) it is necessary to introduce the auxiliary quantity

$$\varepsilon_n(m) \doteq \frac{1}{e^{-m/N}N} \sum_{i=1}^{N} E_D \left[ \delta_{m_i,0} \; Z^{n-2} \int d\mu[f_1] d\mu[f_2] \; \mathcal{L}(f_1, z_i) \; \mathcal{L}(f_2, z_i) \times \right.$$
$$\left. \times \exp\left( -\sum_{j=1}^{N} m_j (h(f_1, z_j) + h(f_2, z_j)) \right) \right] \qquad (9)$$

for $n$ real, which allows to write $\varepsilon(m) = \lim_{n \to 0} \varepsilon_n(m)$. The advantage of this definition is that for *integers* $n > 2$, $\varepsilon_n(m)$ can be represented in terms of $n$ *replicas* of the original variable $f$ for which an explicit average over $m_i$'s is possible. At the end of all calculations an analytical continuation to arbitrary real $n$ and the limit $n \to 0$ must be performed. Using the definition of the partition function (4), we get for integer $n > 2$

$$\varepsilon_n(m) = \frac{1}{e^{-m/N}N} \sum_{i=1}^{N} E_D \left[ \delta_{m_i,0} \int \prod_{a=1}^{n} d\mu[f_a] \; \mathcal{L}(f_1, z_i) \; \mathcal{L}(f_2, z_i) \times \right. \qquad (10)$$
$$\left. \times \exp\left( -\sum_{j=1}^{N} m_j \sum_{a=1}^{n} h(f_a, z_j) \right) \right] .$$

Exchanging the expectation over datasets with the expectation over $f$'s and using the explicit form of the distribution (8) we obtain

$$\varepsilon_n(m) = \Xi_n \frac{1}{e^{-m/N}N} \sum_{i=1}^{N} \left\langle \exp\left[ -\frac{m}{N} \prod_{a=1}^{n} e^{-h(f_a, z_i)} \right] \mathcal{L}(f_1, z_i) \; \mathcal{L}(f_2, z_i) \right\rangle \qquad (11)$$

where the brackets $\langle \ldots \rangle$ denote an average with respect to a Gibbs measure for replicas which is given by

$$p_n(f_1, \ldots, f_n) = \frac{1}{\Xi_n} \prod_{a=1}^{n} \mu[f_a] \; \exp[-H_n] \qquad (12)$$

where

$$H_n = m - \frac{m}{N} \sum_{j=1}^{N} \prod_{a=1}^{n} e^{-h(f_a, z_j)} \qquad (13)$$

and where the partition function $\Xi_n$ has been introduced for convenience to normalize the measure for $n \neq 0$. In most nontrivial cases, averages with respect to the measure (12) can not be calculated exactly. Hence, we have to apply a sensible approximation. Our idea is to use techniques which have been frequently applied to probabilistic models [10] such as the variational approximation, the mean field approximation and the TAP approach. In this paper, we restrict ourself to a variational Gaussian approximation. More advanced approximations will be given elsewhere.

## 5 Variational approximation

A method, frequently used in Statistical Physics which has also attracted considerable interest in the Machine Learning community, is the *variational approximation* [8]. Its goal is to replace an intractable distribution like (12) by a different, sufficiently close distribution from a *tractable class* which we will write in the form

$$p_n^0 \propto \prod_{a=1}^{n} \mu[f_a] \; \exp[-H_n^0] . \qquad (14)$$

$p_n^0$ will be used in (11) instead of $p_n$ to approximate the average. $H_n^0$ will be chosen (see e.g. [10]) to minimize the relative entropy between $p_n^0$ and $p_n$ resulting in a minimization of the *variational free energy*

$$\mathcal{F}_n \doteq -\ln \int \prod_{a=1}^{n} d\mu[f_a] \exp[-H_n^0] + \langle H_n - H_n^0 \rangle_0 \tag{15}$$

being an upper bound to the true free energy $-\ln \Xi_n$ for any integer $n$. The brackets $\langle \ldots \rangle_0$ denote averages with respect to the variational distribution (14).

For our application to Gaussian process models, we will now specialize to Gaussian priors $\mu[f]$. For $H_n^0$, we choose the quadratic expression

$$H_n^0 = \frac{1}{N} \sum_{j=1}^{N} \left( \frac{1}{2} \sum_{a,b=1}^{n} \hat{q}_{ab}(x_j) f_a(x_j) f_b(x_j) + \sum_{a=1}^{n} \hat{r}_a(x_j) f_a(x_j) \right) \tag{16}$$

as a suitable trial Hamiltonian, leading to a Gaussian distribution (14). The functions $\hat{q}_{ab}(x_j)$ and $\hat{r}_a(x_j)$ are the variational parameters to be optimized. To continue the variational solutions to arbitrary real $n$, we assume that the optimal parameters should be replica symmetric, i.e. we set $\hat{r}_a(x_j) = \hat{r}(x_j)$ as well as $\hat{q}_{ab}(x_j) = \hat{q}(x_j)$ for $a \neq b$ and $\hat{q}_{aa}(x_j) = \hat{q}_0(x_j)$. The variational free energy can then be expressed by the local moments ("order parameters" in the language of Statistical Physics) $R(x_j) \doteq \langle f_a(x_j) \rangle_0$, $V(x_l, x_j) \doteq \langle f_a(x_l) f_b(x_j) \rangle_0 - \langle f_a(x_l) \rangle_0 \langle f_b(x_j) \rangle_0$ for $a \neq b$ and $C(x_l, x_j) \doteq \langle f_a(x_l) f_a(x_j) \rangle_0 - \langle f_a(x_l) f_b(x_j) \rangle_0$ which have the same replica symmetric structure. Since each of the $n \times n$ matrices (such as $\hat{q}_{ab}$) are assumed to have only two types of entries, it is possible to obtain variational equations which contain the number $n$ of replicas as a simple parameter for which the limit $n \to 0$ can be explicitly performed (see appendix). In this limit, the limiting order parameters $R(x_j)$, $V(x_j, x_j)$ are found to have simple interpretations as the (approximate) mean and variance of the predictor $\hat{f}_D(x_j)$ with respect to the average over bootstrap data sets while $C(x_l, x_j)$ becomes the (approximate) bootstrap averaged posterior covariance.

## 6 Explicit results for regression with Gaussian processes

We consider a GP model for regression with training energy given by Eq. (2). In this case, the prior measure $\mu[f]$ can be simply represented by an $N$ dimensional Gaussian distribution for the vector $(f(x_1), \ldots, f(x_N))$ having zero mean and covariance matrix $K(x_i, x_j)$, where $K(x, y)$ is the covariance *kernel* of the GP.

Using the limiting (for $n \to 0$) values of order parameters, and by approximating $p_n$ by $p_n^0$ in Eq.(11), the explicit result for the bootstrap mean square generalization error is found to be

$$\varepsilon(m) = \frac{1}{e^{-\mu} N} \sum_{i=1}^{N} [(R(x_i) - y_i)^2 + V(x_i, x_i)] \sum_{k=0}^{\infty} \frac{(-\mu)^k}{k!} \frac{1}{(1 + kC(x_i, x_i)/\sigma^2)^2} \cdot \tag{17}$$

The entire analysis can be repeated for testing (keeping the training energy fixed) with a general loss function of the type $g(\hat{f}_D(x_i) - y)$. The result is

$$\begin{aligned} \varepsilon^{(g)}(m) &= \frac{1}{e^{-\mu} N} E_D \left[ \sum_{i=1}^{N} \delta_{m_i, 0} \, g(\hat{f}_D(x_i) - y_i) \right] \\ &= \frac{1}{e^{-\mu} N} \sum_{i=1}^{N} \int \frac{dv \, e^{-v^2/2}}{\sqrt{2\pi}} \sum_{k=0}^{\infty} \frac{(-\mu)^k}{k!} g \left( \frac{R(x_i) - y_i + v\sqrt{V(x_i, x_i)}}{1 + kC(x_i, x_i)/\sigma^2} \right) \cdot \end{aligned} \tag{18}$$

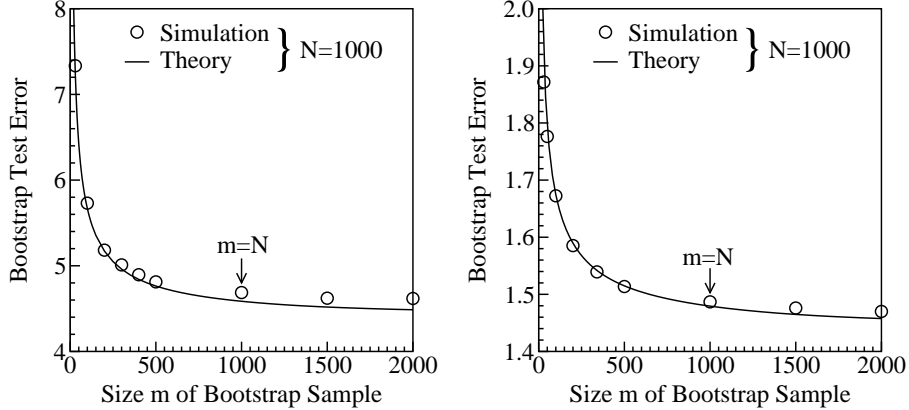

Figure 1: Average bootstrapped generalization error on Abalone data using square error loss (left) and epsilon insensitive loss (right). Simulation (circles) and theory (lines) based on the same data set $D_0$ with $N = 1000$ data points. The GP model uses an RBF kernel $K(x, x') = \exp(-||x - x'||^2/2l^2)$ with $l^2 = 5$ on whitened inputs. For the data noise we set $\sigma^2 = 0.1$.

We have applied our theory to the *Abalone data set* [11] where we have computed the approximate bootstrapped generalization errors for the square error loss and the so-called $\epsilon$-*insensitive loss* which is defined by

$$g(\delta) = \begin{cases} 0 & \text{if} \quad |\delta| \in [0, (1-\beta)\epsilon] \\ \frac{(|\delta| - (1-\beta)\epsilon)^2}{4\beta\epsilon} & \text{if} \quad |\delta| \in [(1-\beta)\epsilon, (1+\beta)\epsilon] \\ |\delta| - \epsilon & \text{if} \quad |\delta| \in [(1+\beta)\epsilon, \infty] \end{cases} \quad (19)$$

with $\delta := \hat{f}_D(x_i) - y_i$. We have set $\beta = 0.9$ and $\epsilon = 0.1$. The bootstrap average from our theory is obtained from Eq.(18). Figure 1 shows the generalization error measured by the square error loss (Eq.(17), left panel) as well as the one measured by the $\epsilon$-insensitive loss (right panel). Our theory (line) is compared with simulations (circles) which were based on Monte-Carlo sampling averages that were computed using the same data set $D_0$ having $N = 1000$. The Monte-Carlo training sets $D$ of size $m$ are obtained by sampling from $D_0$ with replacement. We find a good agreement between theory and simulations in the region were $m < N$. When we oversample the data set $m > N$, however, the agreement is not so good and corrections to our variational Gaussian approximation would be required.

Figure 2 shows the bootstrap average of the posterior variance $\frac{1}{N} \sum_{i=1}^{N} C(x_i, x_i)$ over the whole data set $D_0$, $N = 1000$, and compares our theory (line) with simulations (circles) which were based on Monte-Carlo sampling averages. The overall approximation looks better than for the bootstrap generalization error.

Finally, it is important to note that all displayed theoretical learning curves have been obtained computationally much faster than their respective simulated learning curves.

## 7   Outlook

The replica approach to bootstrap averages can be extended in a variety of different directions. Besides the average generalization error, one can compute its bootstrap *sample fluctuations* by introducing more complicated replica expressions. It is also straightforward to apply the approach to more complex problems in supervised learning which are related to Gaussian processes, such as GP classifiers or Support-vector Machines. Since

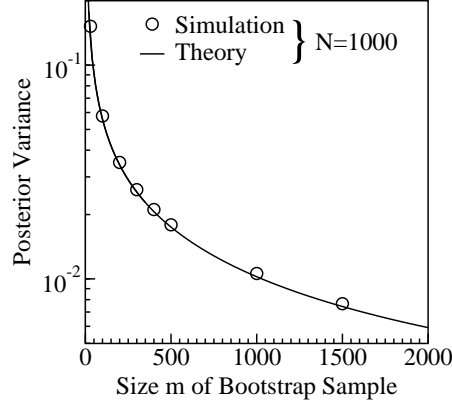

Figure 2: Bootstrap averaged posterior variance for Abalone data. Simulation (circles) and theory (line) based on the same data set $D_0$ with $N = 1000$ data points.

our method requires the solution of a set of variational equations of the size of the original training set, we can expect that its computational complexity should be similar to the one needed for making the actual predictions with the basic model. This will also apply to the problem of very large datasets, where one may use a variety of well known *sparse approximations* (see e.g. [9] and references therein). It will also be important to assess the quality of the approximation introduced by the variational method and compare it to alternative approximation techniques in the computation of the replica average (11), such as the mean field method and its more complex generalizations (see e.g. [10]).

## Acknowledgement

We would like to thank Lars Kai Hansen for stimulating discussions. DM thanks the Copenhagen Image and Signal Processing Graduate School for financial support.

## Appendix: Variational equations

For reference, we will give the explicit form of the equations for variational and order parameters in the limit $n \to 0$. The derivations will be given elsewhere. We obtain

$$R(x_i) = -\frac{1}{N} \sum_{j=1}^{N} C(x_i, x_j) \hat{r}(x_j) \tag{20}$$

$$V(x_i, x_k) = -\frac{1}{N} \sum_{j=1}^{N} C(x_i, x_j) C(x_k, x_j) \hat{q}(x_j) \tag{21}$$

where the matrix $C(x_i, x_j)$ is given by

$$C = \left( K^{-1} + u \right)^{-1} \tag{22}$$

where $K_{ij} = K(x_i, x_j)$ is the kernel matrix. Finally $u_{ij} = \frac{1}{N} \delta_{ij} (\hat{q}_0(x_i) - \hat{q}(x_i))$.

The order parameter equations Eqs.(20-22) must be solved together with the variational equations which are given by

$$\Delta \hat{q}(x_i) = \frac{m}{(\sigma^2 + C(x_i, x_i))} \tag{23}$$

$$\hat{r}(x_i) = -y_i \Delta \hat{q}(x_i) \tag{24}$$

$$\hat{q}(x_i) = -[(R(x_i) - y_i)^2 + V(x_i, x_i)]\frac{(\Delta \hat{q}(x_i))^2}{m} \tag{25}$$

with $\Delta \hat{q}(x_i) = (\hat{q}_0(x_i) - \hat{q}(x_i))$.

Combining Eqs.(22) and (23), a self consistent matrix equation $C = (I + Ku)^{-1}K$ is obtained where $u$ depends on the diagonal elements $C(x_i, x_i)$. Its iterative solution (based on a good initial guess for $C(x_i, x_i)$) requires usually only a few iterations. The order parameters $R(x_i)$ and $V(x_i, x_i)$ can then be solved subsequently using Eq.(20,21) with (24,25).

## Footnotes

[1]The average is over the unknown distribution of training data sets.

## References

[1] A. Engel and C. Van den Broeck, *Statistical Mechanics of Learning* (Cambridge University Press, 2001).

[2] H. Nishimori, *Statistical Physics of Spin Glasses and Information Processing* (Oxford Science Publications, 2001).

[3] B. Efron and R. J. Tibshirani, *An Introduction to the Bootstrap*, Monographs on Statistics and Applied Probability 57 (Chapman&Hall, 1993).

[4] M. Mézard, G. Parisi, and M. A. Virasoro, *Spin Glass Theory and Beyond*, Lecture Notes in Physics **9** (World Scientific, 1987).

[5] J. Shao and D. Tu, *The Jackknife and Bootstrap*, Springer Series in Statistics (Springer Verlag, 1995).

[6] D. Malzahn and M. Opper, *A variational approach to learning curves*, NIPS **14**, Editors: T.G. Dietterich, S. Becker, Z. Ghahramani, (MIT Press, 2002).

[7] R. Neal, *Bayesian Learning for Neural Networks*, Lecture Notes in Statistics **118** (Springer, 1996).

[8] R. P. Feynman and A. R. Hibbs, *Quantum mechanics and path integrals* (Mc Graw-Hill Inc., 1965).

[9] L. Csató and M. Opper, *Sparse Gaussian Processes*, Neural Computation 14, No 3, 641 - 668 (2002).

[10] M. Opper and D. Saad (editors), *Advanced Mean Field Methods: Theory and Practice*, (MIT Press, 2001).

[11] From `http://www1.ics.uci.edu/~mlearn/MLSummary.html`. The data set contains 4177 examples. We used a representative fraction (the forth block (a 1000 data) from the list).
